# The Computation of Sound Source Elevation in the Barn Owl

**Clay D. Spence**
**John C. Pearson**
David Sarnoff Research Center
CN5300
Princeton, NJ 08543-5300

## ABSTRACT

The midbrain of the barn owl contains a map-like representation of sound source direction which is used to precisely orient the head toward targets of interest. Elevation is computed from the interaural difference in sound level. We present models and computer simulations of two stages of level difference processing which qualitatively agree with known anatomy and physiology, and make several striking predictions.

## 1    INTRODUCTION

The auditory system of the barn owl constructs a map of sound direction in the external nucleus of the inferior colliculus (ICx) after several stages of processing the output of the cochlea. This representation of space enables the owl to orient its head to sounds with an accuracy greater than any other tested land animal [Knudsen, et al, 1979]. Elevation and azimuth are processed in separate streams before being merged in the ICx [Konishi, 1986]. Much of this processing is done with neuronal maps, regions of tissue in which the position of active neurons varies continuously with some parameters, e.g., the retina is a map of spatial direction. In this paper we present models and simulations of two of the stages of elevation processing that make several testable predictions. The relatively elaborate structure of this system emphasizes one difference between the sum-and-sigmoid model neuron and real neurons, namely the difficulty of doing subtraction with real neurons. We first briefly review the available data on the elevation system.

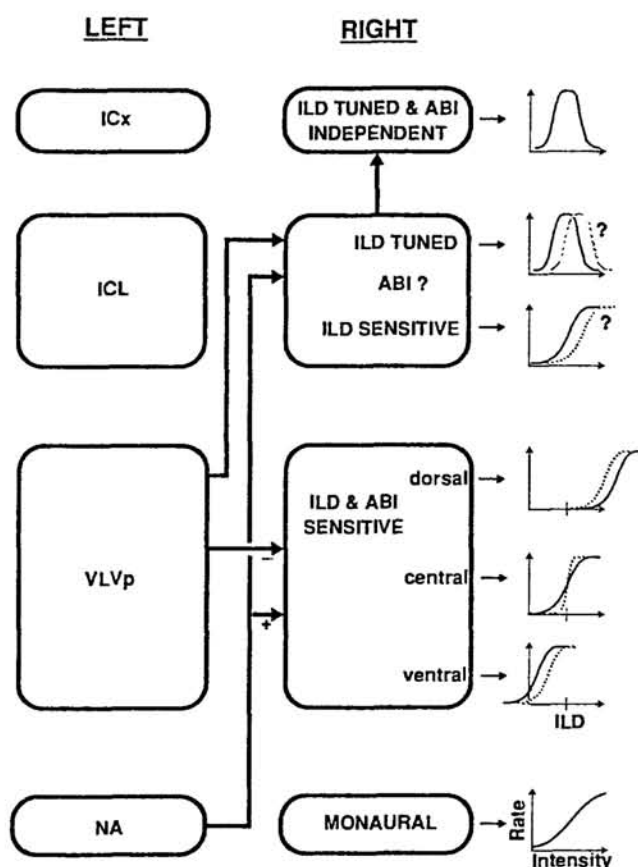

**Figure 1:** Overview of the Barn Owl's Elevation System. ABI: average binaural intensity. ILD: Interaural level difference. Graphs show cell responses as a function of ILD (or monaural intensity for NA).

## 2    KNOWN PROPERTIES OF THE ELEVATION SYSTEM

The owl computes the elevation to a sound source from the inter-aural sound pressure level difference (ILD).[1] Elevation is related to ILD because the owl's ears are asymmetric, so that the right ear is most sensitive to sounds from above, and the left ear is most sensitive to sounds from below [Moiseff, 1989].

After the cochlea, the first nucleus in the ILD system is nucleus angularis (NA) (Fig. 1). NA neurons are monaural, responding only to ipsilateral stimuli.[2] Their outputs are a simple spike rate code for the sound pressure level on that side of the head, with firing rates that increase monotonically with sound pressure level over a rather broad range, typically 30 dB [Sullivan and Konishi, 1984].

Each NA projects to the contralateral nucleus ventralis lemnisci lateralis pars posterior (VLVp). VLVp neurons are excited by contralateral stimuli, but inhibited by ipsilateral stimuli. The source of the ipsilateral inhibition is the contralateral VLVp [Takahashi, 1988]. VLVp neurons are said to be sensitive to ILD, that is their ILD response curves are sigmoidal, in contrast to ICx neurons which are said to be tuned to ILD, that is their ILD response curves are bell-shaped. Frequency is mapped along the anterior-posterior direction, with slabs of similarly tuned cells perpendicular to this axis. Within such a slab, cell responses to ILD vary systematically along the dorsal-ventral axis, and show no variation along the medio-lateral axis. The strength of ipsilateral inhibition[3] varies roughly sigmoidally along the dorsal-ventral axis, being nearly 100% dorsally and nearly 0% ventrally. The ILD threshold, or ILD at which the cell's response is half its maximum value, varies from about 20 dB dorsally to −20 dB ventrally. The response of these neurons is not independent of the average binaural intensity (ABI), so they cannot code elevation unambiguously. As the ABI is increased, the ILD response curves of dorsal cells shift to higher ILD, those of ventral cells shift to lower ILD, and those of central cells keep the same thresholds, but their slopes increase (Fig. 1) [Manley, et al, 1988].

Each VLVp projects contralaterally to the lateral shell of the central nucleus of the inferior colliculus (ICL) [T. T. Takahashi and M. Konishi, unpublished]. The ICL appears to be the nucleus in which azimuth and elevation information is merged before forming the space map in the ICx [Spence, et al, 1989]. At least two kinds of ICL neurons have been observed, some with ILD-sensitive responses as in the VLVp and some with ILD-tuned responses as in the ICx [Fujita and Konishi, 1989]. Manley, Köppl and Konishi have suggested that inputs from both VLVps could interact to form the tuned responses [Manley, et al, 1988]. The second model we will present suggests a simple method for forming tuned responses in the ICL with input from only one VLVp.

## 3    A MODEL OF THE VLVp

We have developed simulations of matched iso-frequency slabs from each VLVp in order to investigate the consequences of different patterns of connections between them. We attempted to account for the observed gradient of inhibition by using a gradient in the number of inhibitory cells. A dorsal-ventral gradient in the number density of different cell types has been observed in staining experiments [C. E. Carr, et al, 1989], with GABAergic cells[4] more numerous at the dorsal end and a non-GABAergic type more numerous at the ventral end.

To model this, our simulation has a "unit" representing a group of neurons at each of forty positions along the VLVp. Each unit has a voltage $v$ which obeys the equation

$$C\frac{dv}{dt} = -g_L(v - v_L) - g_E(v - v_E) - g_I(v - v_I).$$

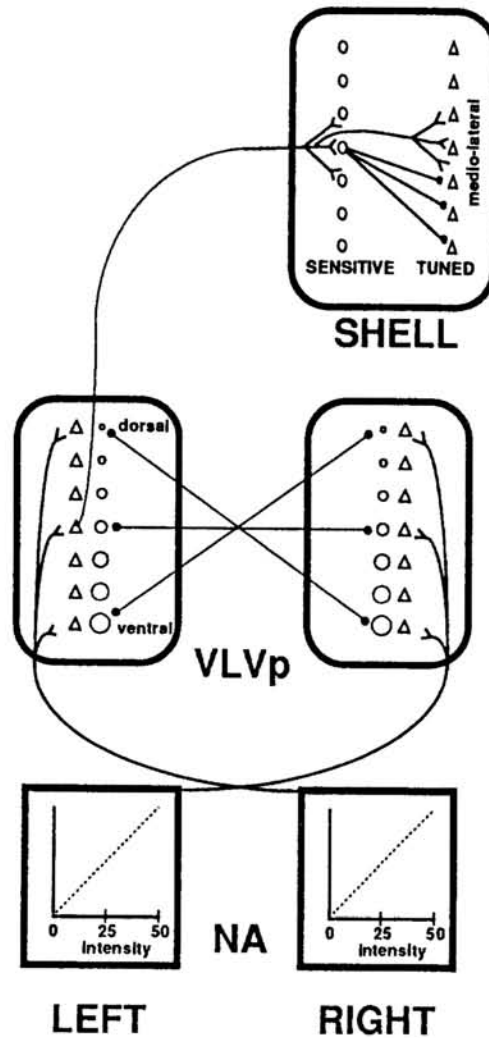

**Figure 2:** Models of Level Difference Computation in the VLVps and Generation of Tuned Responses in the ICL. Sizes of Circles represent the number density of inhibitory neurons, while triangles represent excitatory neurons.

This describes the charging and discharging of the capacitance $C$ through the various conductances $g$, driven by the voltages $v_N$, all of these being properties of the cell membrane. The subscript $L$ refers to passive leakage variables, $E$ refers to excitatory variables, and $I$ refers to inhibitory variables. These model units have firing rates which are sigmoidal functions of $v$. The output on a given time step is a number of spikes, which is chosen randomly with a Poisson distribution whose mean is the unit's current firing rate times the length of the time step. $g_E$ and $g_I$ obey the equation

$$\frac{d^2g}{dt^2} = -\gamma \frac{dg}{dt} - \omega^2 g,$$

the equation for a damped harmonic oscillator. The effect of one unit's spike on another unit is to "kick" its conductance $g$, that is it simply increments the conductance's time derivative by some amount depending on the strength of the connection.

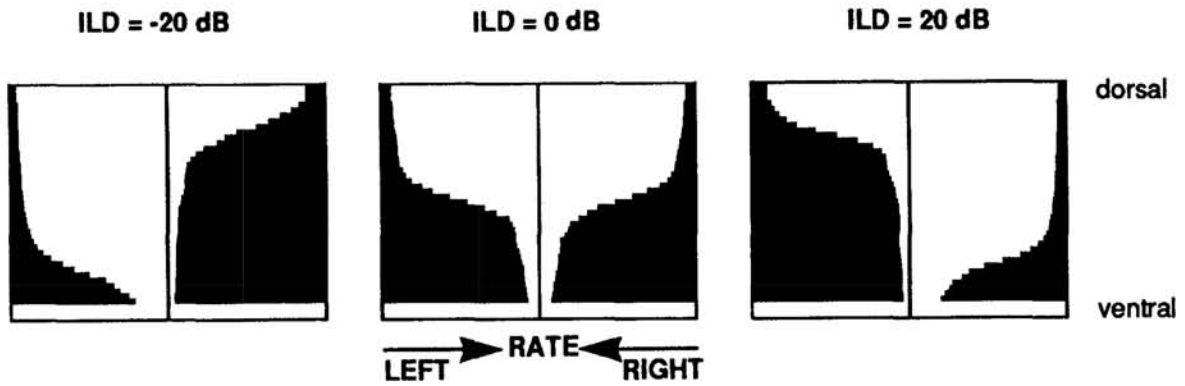

**Figure 3:** Output of Simulation of VLVps at Several ILDs. Position is represented on the vertical axis. Firing rate is represented by the horizontal length of the black bars.

Inhibitory neurons increment $dg_I/dt$, while excitatory neurons increment $dg_E/dt$. $\gamma$ and $\omega$ are chosen so that the oscillator is at least critically damped, and $g$ remains non-negative. This model gives a fairly realistic post-synaptic potential, and the effects of multiple spikes naturally add. The gradient of cell types is modeled by having a different maximum firing rate at each level in the VLVp.

The VLVp model is shown in figure 2. Here, central neurons of each VLVp project to central neurons of the other VLVp, while more dorsal neurons project to more ventral neurons, and conversely. This forms a sort of "criss-cross" pattern of projections. In our simulation these projections are somewhat broad, each unit projecting with equal strength to all units in a small patch. In order for the dorsal neurons to be more strongly inhibited, there must be more inhibitory neurons at the ventral end of each VLVp, so in our simulation the maximum firing rate is higher there and decreases linearly toward the dorsal end. A presumed second neuron type is used for ouput, but we assumed its inputs and dynamics were the same as the inhibitory neurons and so we didn't model them. The input to the VLVps from the two NAs was modeled as a constant input proportional to the sound pressure level in the corresponding ear. We did not use Poisson distributed firing in this case because the spike trains of NA neurons are very regular [Sullivan and Konishi, 1984]. NA input was the same to each unit in the VLVp.

Figure 3 shows spatial activity patterns of the two simulated VLVps for three different ILDs, all at the same ABI. The criss-cross inhibitory connections effectively cause these bars of activity to compete with each other so that their lengths are always approximately complementary. Figure 4 presents results of both models discussed in this paper for various ABIs and ILDs. The output of VLVp units qualitatively matches the experimentally determined responses, in particular the ILD response curves show similar shifts with ABI. for the different dorsal-ventral positions in the VLVp (see Fig. 3 in [Manley, et al, 1988]). Since the observed non-GABAergic neurons are more numerous at the ventral end of the VLVp and

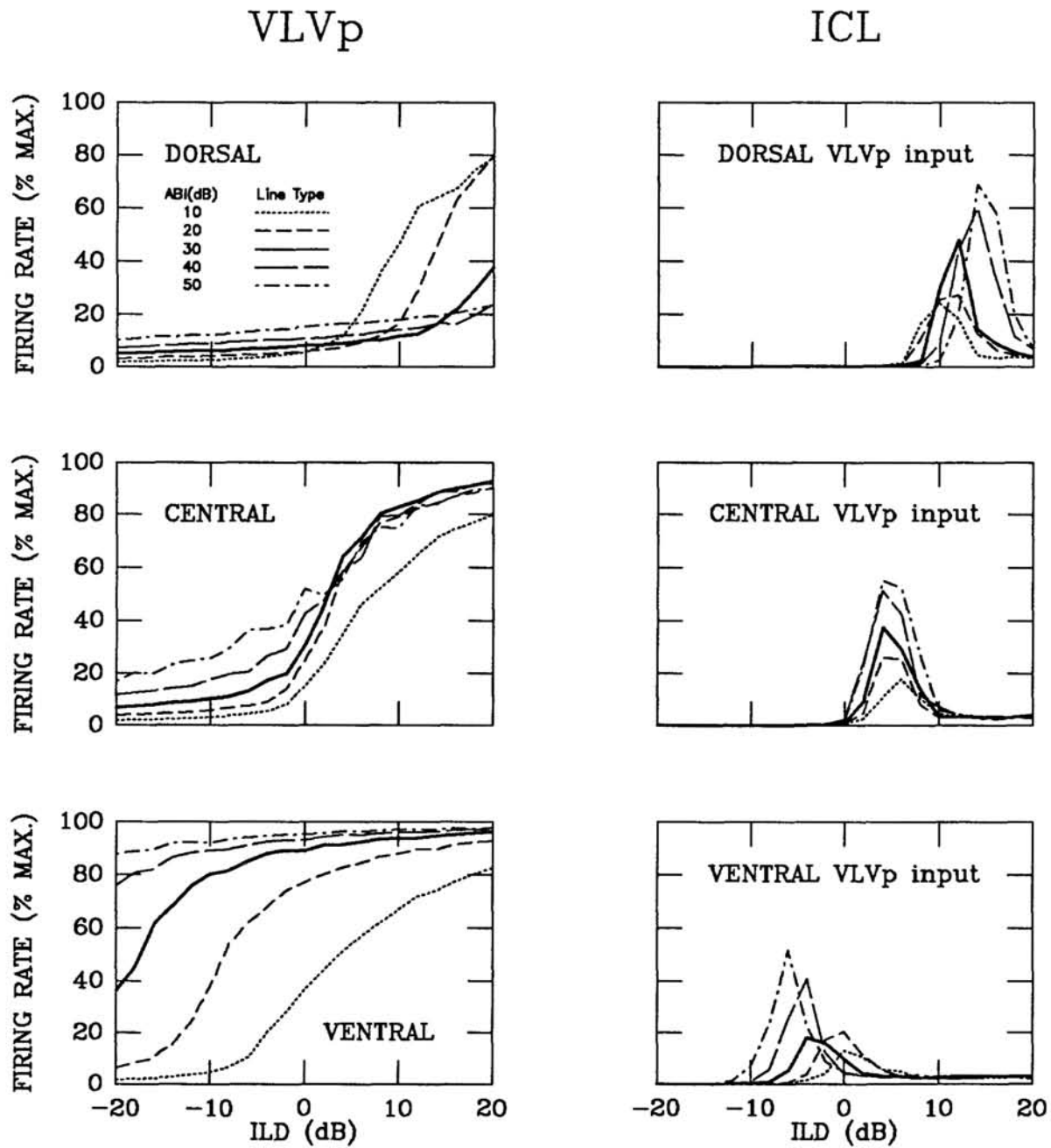

**Figure 4:** ILD Response Curves of the VLVp and ICL models. Curves show percent of maximum firing rate versus ILD for several ABIs.

our model's inhibitory neurons are also more numerous there, this model predicts that at least some of the non-GABAergic cells in the VLVp are the neurons which provide the mutual inhibition between the VLVps.

## 4   A MODEL OF ILD-TUNED NEURONS IN THE ICL

In this section we present a model to explain how ICL neurons can be tuned to ILD if they only receive input from the ILD-sensitive neurons in one VLVp. The model essentially takes the derivative of the spatial activity pattern in the VLVp, converting the sigmoidal activity pattern into a pattern with a localized region of activity corresponding to the end of the bar.

The model is shown in figure 2. The VLVp projects topographically to ICL neurons, exciting two different types. This would excite bars of activity in the ICL, except one type of ICL neuron inhibits the other type. Each inhibitory neuron projects to tuned neurons which represent a smaller ILD, to one side in the map. The inhibitory neurons acquire the bar shaped activity pattern from the VLVp, and are ILD-sensitive as a result. Of the neurons of the second type, only those which receive input from the end of the bar are not also inhibited and prevented from firing.

Our simulation used the model neurons described above, with input to the ICL taken from our model of the VLVp. Each unit in the VLVp projected to a patch of units in the ICL with connection strengths proportional to a gaussian function of distance from the center of the patch. (Equal strengths for the connections from a given neuron worked poorly.) The results are shown in figure 4. The model shows sharp tuning, although the maximum firing rates are rather small. The ILD response curves show the same kind of ABI dependence as those of the VLVp model. There is no published data to confirm or refute this, but we know that neurons in the space map in the ICx do not show ABI dependence. There is a direct input from the contralateral NA to the ICL which may be involved in removing ABI dependence, but we have not considered that possibility in this work.

## 5   CONCLUSION

We have presented two models of parts of the owl's elevation or interaural level difference (ILD) system. One predicts a "criss-cross" geometry for the connections between the owl's two VLVps. In this geometry cells at the dorsal end of either VLVp inhibit cells at the ventral end of the other, and are inhibited by them. Cells closer to the center of one VLVp interact with cells closer to the center of the other, so that the central cells of each VLVp interact with each other (Fig. 2). This model also predicts that the non-GABAergic cells in the VLVp are the cells which project to the other VLVp. The other model explains how the ICL, with input from one VLVp, can contain neurons tuned to ILD. It does this essentially by computing the spatial derivative of the activity pattern in the VLVp. This model predicts that the ILD-sensitive neurons in the ICL inhibit the ILD-tuned neurons in the ICL. Simulations with semi-realistic model neurons show that these models

are plausible, that is they can qualitatively reproduce the published data on the responses of neurons in the VLVp and the ICL to different intensities of sound in the two ears.

Although these are models, they are good examples of the simplicity of information processing in neuronal maps. One interesting feature of this system is the elaborate mechanism used to do subtraction. With the usual model of a neuron, which calculates a sigmoidal function of a weighted sum of its inputs, subtraction would be very easy. This demonstrates the inadequacy of such simple model neurons to provide insight into some real neural functions.

## Acknowledgements

This work was supported by AFOSR contract F49620-89-C-0131.

## Footnotes

[1] Azimuth is computed from the interaural time or phase delay.

[2] Neurons in all of the nuclei we will discuss except ICx have fairly narrow frequency tuning curves.

[3] measured functionally, not actual synaptic strength. See [Manley, et al, 1988] for details.

[4] GABAergic cells are usually thought to be inhibitory.

## References

C. E. Carr, I. Fujita, and M. Konishi. (1989) Distribution of GABAergic neurons and terminals in the auditory system of the barn owl. *The Journal of Comparative Neurology* **286**: 190–207.

I. Fujita and M. Konishi. (1989) Transition from single to multiple frequency channels in the processing of binaural disparity cues in the owl's midbrain. *Society for Neuroscience Abstracts* **15**: 114.

E. I. Knudsen, G. G. Blasdel, and M. Konishi. (1979) Sound localization by the barn owl measured with the search coil technique. *Journal of Comparative Physiology* **133**:1–11.

M. Konishi. (1986) Centrally synthesized maps of sensory space. *Trends in Neurosciences* April, 163–168.

G. A. Manley, C. Köppl, and M. Konishi. (1988) A neural map of interaural intensity differences in the brain stem of the barn owl. *The Journal of Neuroscience* **8**(8): 2665–2676.

A. Moiseff. (1989) Binaural disparity cues available to the barn owl for sound localization. *Journal of Comparative Physiology* **164**: 629–636.

C. D. Spence, J. C. Pearson, J. J. Gelfand, R. M. Peterson, and W. E. Sullivan. (1989) Neuronal maps for sensory-motor control in the barn owl. In D. S. Touretzky (ed.), *Advances in Neural Information Processing Systems 1*, 748–760. San Mateo, CA: Morgan Kaufmann.

W. E. Sullivan and M. Konishi. (1984) Segregation of stimulus phase and intensity coding in the cochlear nucleus of the barn owl. *The Journal of Neuroscience* **4**(7): 1787–1799.

T. T. Takahashi. (1988) Commissural projections mediate inhibition in a lateral lemniscal nucleus of the barn owl. *Society for Neuroscience Abstracts* **14**: 323.